# A Neural Network for Feature Extraction

**Nathan Intrator**
Div. of Applied Mathematics, and
Center for Neural Science
Brown University
Providence, RI 02912

## ABSTRACT

The paper suggests a statistical framework for the parameter estimation problem associated with unsupervised learning in a neural network, leading to an exploratory projection pursuit network that performs feature extraction, or dimensionality reduction.

## 1 INTRODUCTION

The search for a possible presence of some unspecified structure in a high dimensional space can be difficult due to the *curse of dimensionality* problem, namely the inherent sparsity of high dimensional spaces. Due to this problem, uniformly accurate estimations for all smooth functions are not possible in high dimensions with practical sample sizes (Cox, 1984, Barron, 1988).

Recently, exploratory projection pursuit (PP) has been considered (Jones, 1983) as a potential method for overcoming the curse of dimensionality problem (Huber, 1985), and new algorithms were suggested by Friedman (1987), and by Hall (1988, 1989). The idea is to find low dimensional projections that provide the most revealing views of the full-dimensional data emphasizing the discovery of nonlinear effects such as clustering.

Many of the methods of classical multivariate analysis turn out to be special cases of PP methods. Examples are principal component analysis, factor analysis, and discriminant analysis. The various PP methods differ by the projection index optimized.

Neural networks seem promising for feature extraction, or dimensionality reduction, mainly because of their powerful parallel computation. Feature detecting functions of neurons have been studied in the past two decades (von der Malsburg, 1973, Nass et al., 1973, Cooper et al., 1979, Takeuchi and Amari, 1979). It has also been shown that a simplified neuron model can serve as a principal component analyzer (Oja, 1982).

This paper suggests a statistical framework for the parameter estimation problem associated with unsupervised learning in a neural network, leading to an exploratory PP network that performs feature extraction, or dimensionality reduction, of the training data set. The formulation, which is similar in nature to PP, is based on a minimization of a cost function over a set of parameters, yielding an optimal decision rule under some norm. First, the formulation of a single and a multiple feature extraction are presented. Then a new projection index (cost function) that favors directions possessing multimodality, where the multimodality is measured in terms of the separability property of the data, is presented. This leads to the synaptic modification equations governing learning in Bienenstock, Cooper, and Munro (BCM) neurons (1982). A network is presented based on the multiple feature extraction formulation, and both, the linear and nonlinear neurons are analysed.

## 2    SINGLE FEATURE EXTRACTION

We associate a feature with each projection direction. With the addition of a threshold function we can say that an input posses a feature associated with that direction if its projection onto that direction is larger than the threshold. In these terms, a one dimensional projection would be a single feature extraction.

The approach proceeds as follows: Given a compact set of parameters, define a family of loss functions, where the loss function corresponds to a decision made by the neuron whether to fire or not for a given input. Let the risk be the averaged loss over all inputs. Minimize the risk over all possible decision rules, and then minimize the risk over the parameter set. In case the risk does not yield a meaningful minimization problem, or when the parameter set over which the minimization takes place can be restricted by some a-priori knowledge, a penalty, i.e. a measure on the parameter set, may be added to the risk.

Define the decision problem $(\Omega, \mathcal{F}_\Omega, P, L, \mathcal{A})$, where $\Omega = (x^{(1)}, \ldots, x^{(n)})$, $x^{(i)} \in R^N$, is a fixed set of input vectors, $(\Omega, \mathcal{F}_\Omega, P)$ the corresponding probability space, $\mathcal{A} = \{0, 1\}$ the decision space, and $\{L_\theta\}_{\theta \in B^M}$, $L_\theta : \Omega \times \mathcal{A} \mapsto R$ is the family of loss functions. $B^M$ is a compact set in $R^M$. Let $\mathcal{D}$ be the space of all decision rules. The risk $R_\theta : \mathcal{D} \mapsto R$, is given by:

$$R_\theta(\delta) = \sum_{i=1}^{n} P(x^{(i)}) L_\theta(x^{(i)}, \delta(x^{(i)})). \tag{2.1}$$

For a fixed $\theta$, the optimal decision $\delta_\theta$ is chosen so that:

$$R_\theta(\delta_\theta) = \min_{\delta \in \mathcal{D}} \{R_\theta(\delta)\} \tag{2.2}$$

Since the minimization takes place over a finite set, the minimizer exists. In particular, for a given $x^{(i)}$ the decision $\delta_\theta(x^{(i)})$ is chosen so that $L_\theta(x^{(i)}, \delta_\theta(x^{(i)})) \leq L_\theta(x^{(i)}, 1 - \delta_\theta(x^{(i)}))$.

Now we find an optimal $\tilde{\theta}$ that minimizes the risk, namely, $\tilde{\theta}$ will be such that:

$$R_{\tilde{\theta}}(\delta_{\tilde{\theta}}) = \min_{\theta \in B^M} \{R_\theta(\delta_\theta)\}. \tag{2.3}$$

The minimum with respect to $\theta$ exits since $B^M$ is compact.

$R_\theta(\delta_\theta)$ becomes a function that depends only on $\theta$, and when $\theta$ represents a vector in $R^N$, $R_\theta$ can be viewed as a projection index.

# 3     MULTI-DIMENSIONAL FEATURE EXTRACTION

In this case we have a single layer network of interconnected units, each performing a single feature extraction. All units receive the same input and the interaction between the units is via lateral inhibition. The formulation is similar to single feature extraction, with the addition of interaction between the single feature extractors. Let $Q$ be the number of features to be extracted from the data. The multiple decision rule $\delta_\theta = (\delta_\theta^{(1)}, \ldots, \delta_\theta^{(Q)})$ takes values in $\mathcal{A} = \{0,1\}^Q$. The risk of node $k$ is given by: $R_\theta^{(k)}(\delta) = \sum_{i=1}^n P(x^{(i)}) L_\theta^{(k)}(x^{(i)}, \delta^{(k)}(x^{(i)}))$, and the total risk of the network is $R_\theta(\delta) = \sum_{k=1}^Q R_\theta^{(k)}(\delta)$. Proceeding as before, we can minimize over the decision rules $\delta$ to get $\delta_\theta$, and then minimize over $\theta$ to get $\tilde{\theta}$, as in equation (2.3).

The coupling of the equations via the inhibition, and the relation between the different features extracted is exhibited in the loss function for each node and will become clear through the next example.

# 4     FINDING THE OPTIMAL $\theta$ FOR A SPECIFIC LOSS FUNCTION

## 4.1     A SINGLE BCM NEURON - ONE FEATURE EXTRACTION

In this section, we present an exploratory PP method with a specific loss function. The differential equations performing the optimization turn out to be a good approximation of the low governing synaptic weight modification in the BCM theory for learning and memory in neurons. The formal presentation of the theory, and some theoretical analysis is given in (Bienenstock, 1980, Bienenstock et al., 1982), mean field theory for a network based on these neurons is presented in (Scofield and Cooper, 1985, Cooper and Scofield, 1988), more recent analysis based on the statistical viewpoint is in (Intrator 1990), computer simulations and the biological relevance are discussed in (Saul et al., 1986, Bear et al., 1987, Cooper et al., 1988).

We start with a short review of the notations and definitions of BCM theory. Consider a neuron with input vector $x = (x_1, \ldots, x_N)$, synaptic weights vector $m = (m_1, \ldots, m_N)$, both in $R^N$, and activity (in the linear region) $c = x \cdot m$.

Define $\Theta_m = E[(x \cdot m)^2]$, $\hat{\phi}(c, \Theta_m) = c^2 - \frac{2}{3}c\Theta_m$, $\phi(c, \Theta_m) = c^2 - \frac{4}{3}c\Theta_m$. The input $x$, which is a stochastic process, is assumed to be of Type II $\varphi$ mixing, bounded, and piecewise constant. The $\varphi$ mixing property specifies the dependency of the future of the process on its past. These assumptions are needed for the approximation of the resulting deterministic equation by a stochastic one and are discussed in detail in (Intrator, 1990). Note that $c$ represents the linear projection of $x$ onto $m$, and we seek an optimal projection in some sense.

The BCM synaptic modification equations are given by: $\dot{m} = \mu(t)\phi(x \cdot m, \Theta_m)x$, $m(0) = m_0$, where $\mu(t)$ is a global modulator which is assumed to take into account all the global factors affecting the cell, e.g., the beginning or end of the critical period, state of arousal, etc.

Rewriting the modification equation as $\dot{m} = \mu(t)(x \cdot m)(x \cdot m - \frac{4}{3}\theta_m)x$, we see that unlike a classical Hebb-Stent rule, the threshold $\theta_m$ is dynamic. This gives the modification equation the desired stability, with no extra conditions such as saturation of the activity, or normalization of $\| m \|$, and also yields a statistically meaningful optimization.

Returning to the statistical formulation, we let $\theta = m$ be the parameter to be estimated according to the above formulation and define an appropriate loss function depending on the cell's decision whether to fire or not. The loss function represents the intuitive idea that the neuron will fire when its activity is greater than some threshold, and will not otherwise. We denote the firing of the neuron by $a = 1$. Define $K = -\mu \int_{\Theta_m}^{\frac{2}{3}\Theta_m} \hat{\phi}(s, \Theta_m)ds$. Consider the following loss function:

$$L_\theta(x, a) = L_m(x, a) = \begin{cases} -\mu \int_{\Theta_m}^{(x \cdot m)} \hat{\phi}(s, \Theta_m)ds, & (x \cdot m) \geq \Theta_m, \ a = 1 \\ K - \mu \int_{\Theta_m}^{(x \cdot m)} \hat{\phi}(s, \Theta_m)ds, & (x \cdot m) < \Theta_m, \ a = 1 \\ -\mu \int_{\Theta_m}^{(x \cdot m)} \hat{\phi}(s, \Theta_m)ds, & (x \cdot m) \leq \Theta_m, \ a = 0 \\ K - \mu \int_{\Theta_m}^{(x \cdot m)} \hat{\phi}(s, \Theta_m)ds, & (x \cdot m) > \Theta_m, \ a = 0 \end{cases} \quad (4.1)$$

It follows from the definition of $L_\theta$ and from the definition of $\delta_\theta$ in (2.2) that

$$L_m(x, \delta_m) = -\mu \int_{\Theta_m}^{(x \cdot m)} \hat{\phi}(s, \Theta_m)ds = -\frac{\mu}{3}\{(x \cdot m)^3 - E[(x \cdot m)^2](x \cdot m)^2\} \quad (4.2)$$

The above definition of the loss function suggests that the decision of a neuron whether to fire or not is based on a dynamic threshold $(x \cdot m) > \Theta_m$. It turns out that the synaptic modification equations remain the same if the decision is based on a fixed threshold. This is demonstrated by the following loss function, which leads to the same risk as in equation (4.3): $K = -\mu \int_0^{\frac{2}{3}\Theta_m} \hat{\phi}(s, \Theta_m)ds$,

$$L_\theta(x, a) = L_m(x, a) = \begin{cases} -\mu \int_0^{(x \cdot m)} \hat{\phi}(s, \Theta_m)ds, & (x \cdot m) \geq 0, \ a = 1 \\ K - \mu \int_0^{(x \cdot m)} \hat{\phi}(s, \Theta_m)ds, & (x \cdot m) < 0, \ a = 1 \\ -\mu \int_0^{(x \cdot m)} \hat{\phi}(s, \Theta_m)ds, & (x \cdot m) \leq 0, \ a = 0 \\ K - \mu \int_0^{(x \cdot m)} \hat{\phi}(s, \Theta_m)ds, & (x \cdot m) > 0, \ a = 0 \end{cases} \quad (4.1')$$

The risk is given by:

$$R_\theta(\delta_\theta) = -\frac{\mu}{3}\{E[(x \cdot m)^3] - E^2[(x \cdot m)^2]\}. \tag{4.3}$$

The following graph represents the $\phi$ function and the associated loss function $L_m(x, \delta_m)$ of the activity $c$.

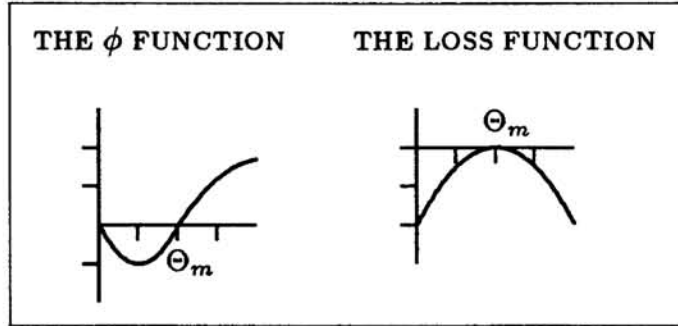

Fig. 1: The Function $\phi$ and the Loss Functions for a Fixed $m$ and $\Theta_m$.

From the graph of the loss function it follows that for any fixed $m$ and $\Theta_m$, the loss is small for a given input $x$, when either $x \cdot m$ is close to zero or negative, or when $x \cdot m$ is larger than $\Theta_m$. This suggests, that the preferred directions for a fixed $\theta_m$ will be such that the projected single dimensional distribution differs from normal in the center of the distribution, in the sense that it has a multi-modal distribution with a distance between the two peaks larger than $\theta_m$. Rewriting (4.3) we get

$$\frac{R_\theta(\delta_\theta)}{E^2[(x \cdot m)^2]} = -\frac{\mu}{3}\{\frac{E[(x \cdot m)^3]}{E^2[(x \cdot m)^2]} - 1\}. \tag{4.4}$$

The term $E[(x \cdot m)^3]/E^2[(x \cdot m)^2]$ can be viewed as some measure of the skewness of the distribution, which is a measure of deviation from normality and therefore an interesting direction (Diaconis and Friedman, 1984), in accordance with Friedman (1987) and Hall's (1988, 1989) argument that it is best to seek projections that differ from the normal in the center of the distribution rather than in the tails.

Since the risk is continuously differentiable, its minimization can be done via the gradient descent method with respect to $m$, namely:

$$\frac{\partial m_i}{\partial t} = -\frac{\partial}{\partial m_i} R_\theta(\delta_\theta) = \mu E[\phi(x \cdot m, \Theta_m)x_i]. \tag{4.5}$$

Notice that the resulting equation represents an averaged deterministic equation of the stochastic BCM modification equations. It turns out that under suitable conditions on the mixing of the input $x$ and the global function $\mu$, equation (4.5) is a good approximation of its stochastic version.

When the nonlinearity of the neuron is emphasized, the neuron's activity is then defined as $c = \sigma(x \cdot m)$, where $\sigma$ usually represents a smooth sigmoidal function. $\Theta_m$ is then defined as $E[\sigma^2(x \cdot m)]$, and the loss function is similar to the one given by equation (4.1) except that $(x \cdot m)$ is replaced by $\sigma(x \cdot m)$. The gradient of

the risk is given by: $-\nabla_m R_m(\delta_m) = \mu E[\phi\Big(\sigma(x \cdot m), \Theta_m\Big)\sigma' x]$, where $\sigma'$ represents the derivative of $\sigma$ at the point $(x \cdot m)$. Note that $\sigma$ may represent any nonlinear function, e.g. radial symmetric kernels.

## 4.2    THE NETWORK - MULTIPLE FEATURE EXTRACTION

In this case we have $Q$ identical nodes, which receive the same input and inhibit each other. Let the neuronal activity be denoted by $c_k = x \cdot m_k$. We define the *inhibited* activity $\tilde{c}_k = c_k - \eta \sum_{j \neq k} c_j$, and the threshold $\tilde{\Theta}_m^k = E[\tilde{c}_k^2]$. In a more general case, the inhibition may be defined to take into account the spatial location of adjacent neurons, namely, $\tilde{c}_k = \sum_j \lambda_{jk} c_j$, where $\lambda_{jk}$ represents different types of inhibitions, e.g. Mexican hat. Since the following calculations are valid for both kinds of inhibition we shall introduce only the simpler one.

The loss function is similar to the one defined in a single feature extraction with the exception that the activity $c = x \cdot m$ is replaced by $\tilde{c}$. Therefore the risk for node $k$ is given by: $R_k = -\frac{\mu}{3}\{E[\tilde{c}_k^3] - (E[\tilde{c}_k^2])^2\}$, and the total risk is given by $R = \sum_{k=1}^{Q} R_k$. The gradient of $R$ is given by:

$$\frac{\partial R}{\partial m_k} = -\mu[1 - \eta(Q - 1)]E[\phi(\tilde{c}_k, \tilde{\Theta}_m^k)x]. \tag{4.6}$$

Equation (4.6) demonstrates the ability of the network to perform exploratory projection pursuit in parallel, since the minimization of the risk involves minimization of nodes $1, \ldots, Q$, which are loosely coupled.

The parameter $\eta$ represents the amount of lateral inhibition in the network, and is related to the amount of correlation between the different features sought by the network. Experience shows that when $\eta \simeq 0$, the different units may all become selective to the simplest feature that can be extracted from the data. When $\eta(Q - 1) \simeq 1$, the network becomes selective to those inputs that are very far apart (under the $l^2$ norm), yielding a classification of a small portion of the data, and mostly unresponsiveness to the rest of the data. When $0 < \eta(Q - 1) < 1$, the network becomes responsive to substructures that may be common to several different inputs, namely extract invariant features in the data. The optimal value of $\eta$ has been estimated by data driven techniques.

When the non linearity of the neuron is emphasized the activity is defined (as in the single neuron case) as $c_k = \sigma(x \cdot m_k)$. $\tilde{c}_k$, $\tilde{\Theta}_m^k$, and $R_k$ are defined as before. In this case $\frac{\partial \tilde{c}_k}{\partial m_j} = -\eta\sigma'(x \cdot m_j)x$, $\frac{\partial \tilde{c}_k}{\partial m_k} = \sigma'(x \cdot m_k)x$, and equation (4.6) becomes:

$$\frac{\partial R}{\partial m_k} = -\mu E\Big[\phi(\tilde{c}_k, \tilde{\Theta}_m^k)\Big(\sigma'(x \cdot m_k) - \eta\sum_{j \neq k}\sigma'(x \cdot m_j)\Big)x\Big] \tag{4.7}$$

## 4.3   OPTIMAL NETWORK SIZE

A major problem in network solutions to real world problems is optimal network size. In our case, it is desirable to try and extract as many features as possible on

one hand, but it is clear that too many neurons in the network will simply inhibit each other, yielding sub-optimal results. The following solution was adopted: We replace each neuron in the network with a group of neurons which all receive the same input, and the same inhibition from adjacent groups. These neurons differ from one another only in their initial synaptic weights. The output of each neuron is replaced by the average group activity. Experiments show that the resulting network is more robust to noise and outliers in the data. Furthermore, it is observed that groups that become selective to a *true* feature in the data, posses a much smaller inter-group variance of their synaptic weight vector than those which do not become responsive to a coherent feature. We found that eliminating neurons with large inter-group variance and retraining the network, may yield improved feature extraction properties.

The network has been applied to speech segments, in an attempt to extract some features from CV pairs of isolated phonemes (Seebach and Intrator, 1988).

## 5 DISCUSSION

The PP method based on the BCM modification function, has been found capable of effectively discovering non linear data structures in high dimensional spaces. Using a parallel processor and the presented network topology, the pursuit can be done faster than in the traditional serial methods.

The projection index is based on polynomial moments, and is therefore computationally attractive. When only the nonlinear structure in the data is of interest, a sphering transformation (Huber, 1981, Friedman, 1987), can be applied first to the data for removal of all the location, scale, and correlational structure from the data.

When compared with other PP methods, the highlights of the presented method are *i*) the projection index concentrates on directions where the separability property as well as the non-normality of the data is large, thus giving rise to better classification properties; *ii*) the degree of correlation between the directions, or features extracted by the network can be regulated via the global inhibition, allowing some tuning of the network to different types of data for optimal results; *iii*) the pursuit is done on all the directions at once thus leading to the capability of finding more interesting structures than methods that find only one projection direction at a time. *iv*) the network's structure suggests a simple method for size-optimization.

### Acknowledgements

I would like to thank Professor Basilis Gidas for many fruitful discussions.

Supported by the National Science Foundation, the Office of Naval Research, and the Army Research Office.

### References

Barron A. R. (1988) Approximation of densities by sequences of exponential families. Submitted to *Ann. Statist.*

Bienenstock E. L. (1980) A theory of the development of neuronal selectivity. Doctoral dissertation, Brown University, Providence, RI

Bienenstock E. L., L. N Cooper, and P. W. Munro (1982) Theory for the development of neuron selectivity: orientation specificity and binocular interaction in visual cortex. *J.Neurosci.* 2:32-48

Bear M. F., L. N Cooper, and F. F. Ebner (1987) A Physiological Basis for a Theory of Synapse Modification. *Science* 237:42-48

Cooper L. N, and F. Liberman, and E. Oja (1979) A theory for the acquisition and loss of neurons specificity in visual cortex. *Biol. Cyb.* 33:9-28

Cooper L. N, and C. L. Scofield (1988) Mean-field theory of a neural network. *Proc. Natl. Acad. Sci. USA* 85:1973-1977

Cox D. D. (1984) Multivariate smoothing spline functions. *SIAM J. Numer. Anal.* 21 789-813

Diaconis P., and D. Freedman (1984) Asymptotics of Graphical Projection Pursuit. *The Annals of Statistics*, 12 793-815.

Friedman J. H. (1987) Exploratory Projection Pursuit. *Journal of the American Statistical Association* 82-397:249-266

Hall P. (1988) Estimating the Direction in which Data set is Most Interesting. *Probab. Theory Rel. Fields* 80, 51-78

Hall P. (1989) On Polynomial-Based Projection Indices for Exploratory Projection Pursuit. *The Annals of Statistics*, 17, 589-605.

Huber P. J. (1981) Projection Pursuit. Research Report PJH-6, Harvard University, Dept. of Statistics.

Huber P. J. (1985) Projection Pursuit. *The Annal. of Stat.* 13:435-475

Intrator N. (1990) An Averaging Result for Random Differential Equations. In Press.

Jones M. C. (1983) The Projection Pursuit Algorithm for Exploratory Data Analysis. Unpublished Ph.D. dissertation, University of Bath, School of Mathematics.

von der Malsburg, C. (1973) Self-organization of orientation sensitivity cells in the striate cortex. *Kybernetik* 14:85-100

Nass M. M., and L. N Cooper (1975) A theory for the development of feature detecting cells in visual cortex. *Biol. Cybernetics* 19:1-18

Oja E. (1982) A Simplified Neuron Model as a Principal Component Analyzer. *J. Math. Biology*, 15:267-273

Saul A., and E. E. Clothiaux, 1986) Modeling and Simulation III: Simulation of a Model for Development of Visual Cortical specificity. *J. of Electrophysiological Techniques*, 13:279-306

Scofield C. L., and L. N Cooper (1985) Development and properties of neural networks. *Contemp. Phys.* 26:125-145

Seebach B. S., and N. Intrator (1988) A learning Mechanism for the Identification of Acoustic Features. (Society for Neuroscience).

Takeuchi A., and S. Amari (1979) Formation of topographic maps and columnar microstructures in nerve fields. *Biol. Cyb.* 35:63-72